# Log-concavity results on Gaussian process methods for supervised and unsupervised learning

**Liam Paninski**
Gatsby Computational Neuroscience Unit
University College London
liam@gatsby.ucl.ac.uk
http://www.gatsby.ucl.ac.uk/~liam

## Abstract

Log-concavity is an important property in the context of optimization, Laplace approximation, and sampling; Bayesian methods based on Gaussian process priors have become quite popular recently for classification, regression, density estimation, and point process intensity estimation. Here we prove that the predictive densities corresponding to each of these applications are log-concave, given any observed data. We also prove that the likelihood is log-concave in the hyperparameters controlling the mean function of the Gaussian prior in the density and point process intensity estimation cases, and the mean, covariance, and observation noise parameters in the classification and regression cases; this result leads to a useful parameterization of these hyperparameters, indicating a suitably large class of priors for which the corresponding maximum *a posteriori* problem is log-concave.

**Introduction**

Bayesian methods based on Gaussian process priors have recently become quite popular for machine learning tasks (*1*). These techniques have enjoyed a good deal of theoretical examination, documenting their learning-theoretic (generalization) properties (*2*), and developing a variety of efficient computational schemes (e.g., (*3–5*), and references therein). We contribute to this theoretical literature here by presenting results on the log-concavity of the predictive densities and likelihood associated with several of these methods, specifically techniques for classification, regression, density estimation, and point process intensity estimation. These results, in turn, imply that it is relatively easy to tune the hyperparameters for, approximate the posterior distributions of, and sample from these models.

Our results are based on methods which we believe will be applicable more widely in machine learning contexts, and so we give all necessary details of the (fairly straightforward) proof techniques used here.

**Log-concavity background**

We begin by discussing the log-concavity property: its uses, some examples of log-concave (l.c.) functions, and the key theorem on which our results are based. Log-concavity is perhaps most important in a maximization context: given a real function $f$ of some vector parameter $\vec{\theta}$, if $g(f(\vec{\theta}))$ is concave for some invertible function $g$, and the parameters $\vec{\theta}$ live in some convex set, then $f$ is unimodal, with no non-global local maxima. (Note that in this case a global maximum, if one exists, is not necessarily unique, but maximizers of $f$ do form a convex set, and hence maxima are essentially unique in a sense.) Thus ascent procedures for maximization can be applied without fear of being trapped in local maxima; this is extremely useful when the space to be optimized over is high-dimensional. This logic clearly holds for any arbitrary rescaling $g$; of course, we are specifically interested in $g(t) = \log t$, since logarithms are useful in the context of taking products (in a probabilistic context, read conditional independence): log-concavity is preserved under multiplication, since the logarithm converts multiplication into addition and concavity is preserved under addition.

Log-concavity is also useful in the context of Laplace (central limit theorem - type) approximations (*3*), in which the logarithm of a function (typically a probability density or likelihood function) is approximated via a second-order (quadratic) expansion about its maximum or mean (*6*); this log-quadratic approximation is a reasonable approach for functions whose logs are known to be concave.

Finally, l.c. distributions are in general easier to sample from than arbitrary distributions, as discussed in the context of adaptive rejection and slice sampling (*7, 8*) and the random-walk-based samplers analyzed in (*9*).

We should note that log-concavity is not a generic property: l.c. probability densities necessarily have exponential tails (ruling out power law tails, and more generally distributions with any infinite moments). Log-concavity also induces a certain degree of smoothness; for example, l.c. densities must be continuous on the interior of their support. See, e.g., (*9*) for more detailed information on the various special properties implied by log-concavity.

A few simple examples of l.c. functions are as follows: the Gaussian density in any dimension; the indicator of any convex set (e.g., the uniform density over any convex, compact set); the exponential density; the linear half-rectifier. More interesting well-known examples include the determinant of a matrix, or the inverse partition function of an energy-based probabilistic model (e.g., an exponential family), $Z^{-1}(\vec{\theta}) = (\int e^{f(\vec{x},\vec{\theta})}d\vec{x})^{-1}$, l.c. in $\vec{\theta}$ whenever $f(\vec{x},\vec{\theta})$ is convex in $\vec{\theta}$ for all $\vec{x}$. Finally, log-concavity is preserved under taking products (as noted above), affine translations of the domain, and/or pointwise limits, since concavity is preserved under addition, affine translations, and pointwise limits, respectively.

Sums of l.c. functions are not necessarily l.c., as is easily shown (e.g., a mixture of Gaussians with widely-separated means, or the indicator of the union of disjoint convex sets). However, a key theorem (*10, 11*) gives:

**Theorem (Integrating out preserves log-concavity).** *If $f(\vec{x},\vec{y})$ is jointly l.c. in $(\vec{x},\vec{y})$, for $\vec{x}$ and $\vec{y}$ finite dimensional, then*

$$f_0(\vec{x}) \equiv \int f(\vec{x},\vec{y})d\vec{y}$$

*is l.c. in $\vec{x}$.*

Think of $\vec{y}$ as a latent variable or hyperparameter we want to marginalize over. This very useful fact has seen applications in various branches of statistics and operations research, but does not seem well-known in the machine learning community. The theorem implies, for example, that convolutions of l.c. functions are l.c.; thus the random vectors

with l.c. densities form a vector space. Moreover, indefinite integrals of l.c. functions are l.c.; hence the error function, and more generally the cumulative distribution function of any l.c. density, is l.c., which is useful in the setting of generalized linear models (*12*) for classification. Finally, the mass under a l.c. probability measure of a convex set which is translated in a convex manner is itself a l.c. function of the convex translation parameter (*11*).

## Gaussian process methods background

We now give a brief review of Gaussian process methods. Our goals are modest; we will do little more than define notation. See, e.g., (*1*) and references for further details. Gaussian process methods are based on a Bayesian "latent variable" approach: dependencies between the observed input and output data $\{\vec{t}_i\}$ and $\{\vec{y}_i\}$ are modeled as arising through a hidden (unobserved) Gaussian process $G(\vec{t})$. Recall that a Gaussian process is a stochastic process whose finite-dimensional projections are all multivariate Gaussian, with means and covariances defined consistently for all possible projections, and is therefore specified by its mean $\mu(\vec{t})$ and covariance function $C(\vec{t}_1, \vec{t}_2)$.

The applications we will consider may be divided into two settings; "supervised" and "unsupervised" problems. We discuss the somewhat simpler unsupervised case first (however, it should be noted that the supervised cases have received significantly more attention in the machine learning literature to date, and might be considered of more importance to this community).

**Density estimation**: We are given unordered data $\{\vec{t}_i\}$; the setup is valid for any sample space, but assume $\vec{t}_i \in \Re^d, d < \infty$, for concreteness. We model the data as i.i.d. samples from an unknown distribution $p$. The prior over these unknown distributions, in turn, is modeled as a conditioned Gaussian process, $p \sim G(\vec{t})$: $p$ is drawn from a Gaussian process $G(\vec{t})$ of mean $\mu(\vec{t})$ and covariance $C$ (to ensure that the resulting random measures are well-defined, we will assume throughout that $G$ is moderately well-behaved; almost-sure local Lebesgue integrability is sufficient), conditioned to be nonnegative and to integrate to one over some arbitrarily large compact set (the latter by an obvious limiting argument, to prevent conditioning on a set of measure zero; the introduction of the compact set is to avoid problems of the sort encountered when trying to define uniform probability measures on unbounded spaces) with respect to some natural base measure on the sample space (e.g., Lebesgue measure in $\Re^d$) (*13*). It is worth emphasizing that this setup differs somewhat from some earlier proposals (*5, 14, 15*), which postulated that nonnegativity be enforced by, e.g., modeling $\log p$ or $\sqrt{p}$ as Gaussian, instead of the Gaussian $p$ here; each approach has its own advantages, and it is unclear at the moment whether our results can be extended to this context (as will be clear below, the roadblock is in the normalization constraint, which is transformed nonlinearly along with the density in the nonlinear warping setup).

**Point process intensity estimation**: A nearly identical setup can be used if we assume the data $\{\vec{t}_i\}$ represent a sample from a Poisson process with an unknown underlying intensity function (*16–18*); the random density above is simply replaced by the random intensity function here (this type of model is known as a Cox, or doubly-stochastic, process in the point-process literature). The only difference is that intensity functions are not required to be normalized, so we need only condition the Gaussian process $G(\vec{t})$ from which we draw the intensity functions to be nonnegative. It turns out we will be free to use any l.c. and convex warping of the range space of the Gaussian process $G(\vec{t})$ to enforce positivity; suitable warpings include exponentiation (corresponding to modeling the logarithm of the intensity as Gaussian (*17*)) or linear half-rectification.

The supervised cases require a few extra ingredients. We are given paired data, inputs $\{\vec{t}_i\}$

with corresponding outputs $\{\vec{y}_i\}$. We model the outputs as noise-corrupted observations from the Gaussian process $\vec{G}(\vec{t})$ at the points $\{\vec{t}_i\}$; denote the additional hidden "observation" noise process as $\{\vec{n}(\vec{t}_i)\}$. This noise process is not always taken to be Gaussian; for computational reasons, $\{\vec{n}(\vec{t}_i)\}$ is typically assumed i.i.d., and also independent of $\vec{G}(\vec{t})$, but both of these assumptions will be unnecessary for the results stated below.

**Regression**: We assume $\vec{y}(\vec{t}_i) = \vec{G}(\vec{t}_i) + \sigma_i \vec{n}(\vec{t}_i)$; in words, draw $\vec{G}(\vec{t})$ from a Gaussian process of mean $\vec{\mu}(\vec{t})$ and covariance $C$; the outputs are then obtained by sampling this function $\vec{G}(\vec{t})$ at $\vec{t}_i$ and adding noise $\vec{n}(\vec{t}_i)$ of scale $\sigma_i$.

**Classification**: $y(\vec{t}_i) = 1\left(G(\vec{t}_i) + \sigma_i n(\vec{t}_i) > 0\right)$, where $1(.)$ denotes the indicator function of an event. This case is as in the regression model, except we only observe a binary-thresholded version of the real output.

## Results

Our first result concerns the predictive densities associated with the above models: the posterior density of any continuous linear functional of $G(\vec{t})$, given observed data $D = \{\vec{t}_i\}$ and/or $\{y_i\}$, under the Gaussian process prior for $G(\vec{t})$. The simplest and most important case of such a linear projection is the projection onto a finite collection of coordinates, $\{\vec{t}_{pred}\}$, say; in this special case, the predictive density is the posterior density $p(\{G(\vec{t}_{pred})\}|D)$. It turns out that all we need to assume is the log-concavity of the distribution $p(G, \vec{n})$; this is clearly more general than what is needed for the strictly Gaussian cases considered above (for example, Laplacian priors on $G$ are permitted, which could lead to more robust performance). Also note that dependence of $(G, \vec{n})$ is allowed; this permits, for example, coupling of the effective scales of the observation noise $\vec{n}_i = \vec{n}(\vec{t}_i)$ for nearby points $\vec{t}_i$. Additonally, we allow nonstationarity and anisotropic correlations in $G$. The result applies for any of the applications discussed above.

**Proposition 1 (Predictive density).** *Given a l.c. prior on $(G, \vec{n})$, the predictive density is always l.c., for any data $D$.*

In other words, conditioning on data preserves these l.c. processes (where an l.c. process, like a Gaussian process, is defined by the log-concavity of its finite-dimensional projections). This represents a significant generalization of the obvious fact that in the regression setup under Gaussian noise, conditioning preserves Gaussian processes.

Our second result applies to the likelihood of the hyperparameters corresponding to the above applications: the mean function $\mu$, the covariance function $C$, and the observation noise scales $\{\sigma_i\}$. We first state the main result in some generality, then provide some useful examples and interpretation below. For each $j > 0$, let $A_{j,\vec{\theta}}$ denote a family of linear maps from some finite-dimensional vector space $\mathcal{G}_j$ to $\Re^{Nd_G}$, where $d_G = \dim(\vec{G}(\vec{t}_i))$, and $N$ is the number of observed data points. Our main assumptions are as follows: first, assume $A_{j,\vec{\theta}}^{-1}$ may be written $A_{j,\vec{\theta}}^{-1} = \sum \theta_k K_{j,k}$, where $\{K_{j,k}\}$ is a fixed set of matrices and the inverse is defined as a map from $\text{range}(A_{j,\vec{\theta}})$ to $\mathcal{G}_j/\ker(A_{j,\vec{\theta}})$. Second, assume that $\dim(A_{j,\vec{\theta}}^{-1}(V))$ is constant in $\vec{\theta}$ for any set $V$. Finally, equip the (doubly) latent space $\mathcal{G}_j \times \Re^{Nd_G} = \{(G_L, \vec{n})\}$ with a translation family of l.c. measures $p_{j,\mu_L}(G_L, \vec{n})$ indexed by the mean parameter $\mu_L$, i.e., $p_{j,\mu_L}(G_L, \vec{n}) = p_j((G_L, \vec{n}) - \mu_L)$, for some fixed measure $p_j(.)$. Then if the sequence $p_j(G, \vec{n})$ induced by $p_j$ and $A_j$ converges pointwise to the joint density $p(G, \vec{n})$, then:

**Proposition 2 (Likelihood).** *In the supervised cases, the likelihood is jointly l.c. in the latent mean function, covariance parameters, and inverse noise scales $(\mu_L, \vec{\theta}, \{\sigma_i^{-1}\})$, for*

*all data D. In the unsupervised cases, the likelihood is l.c. in the mean function $\mu$.*

Note that the mean function $\mu(\vec{t})$ is induced in a natural way by $\mu_L$ and $A_{i,\vec{\theta}}$, and that we allow the noise scale parameters $\{\sigma_i\}$ to vary independently, increasing the robustness of the supervised methods (*19*) (since outliers can be "explained," without large perturbations of the underlying predictive distributions of $G(\vec{t})$, by simply increasing the corresponding noise scale $\sigma_i$). Of course, in practice, it is likely that to avoid overfitting one would want to reduce the effective number of free parameters by representing $\mu(\vec{t})$ and $\vec{\theta}$ in finite-dimensional spaces, and restricting the freedom of the inverse noise scales $\{\sigma_i\}$. The log-concavity in the mean function $\mu(\vec{t})$ demonstrated here is perhaps most useful in the point process setting, where $\mu(\vec{t})$ can model the effect of excitatory or inhibitory inputs on the intensity function, with spatially- or temporally-varying patterns of excitation, and/or self-excitatory interactions between observation sites $\vec{t}_i$ (by letting $\mu(\vec{t})$ depend on the observed points $\vec{t}_i$ (*16, 20*)).

In the special case that the l.c. prior measure $p_j$ is taken to be Gaussian with covariance $C_0$, the main assumption here is effectively on the parameterization of the covariance $C$; ignoring the (technical) limiting operation in $j$ for the moment, we are assuming roughly that there exists a single basis in which, for all allowed $\vec{\theta}$, the covariance may be written $C = A_{\vec{\theta}} C_0 A_{\vec{\theta}}^t$, where $A_{\vec{\theta}}$ is of the special form described above.

We may simplify further by assuming that $C_0$ is white and stationary. One important example of a suitable two-parameter family of covariance kernels satisfying the conditions of Proposition 2 is provided by the Ornstein-Uhlenbeck kernels (which correspond to exponentially-filtered one-dimensional white noise):

$$C(t_1, t_2) = \sigma^2 e^{-2|t_1 - t_2|/\tau}$$

For this kernel, one can parameterize $C = A_{\vec{\theta}} A_{\vec{\theta}}^t$, with $A_{\vec{\theta}}^{-1} = \theta_1 I - \theta_2 D^*$, where $I$ and $D$ denote the identity and differential operators, respectively, and $\theta_k > 0$ to ensure that $C$ is positive-definite. (To derive this reparameterization, note that $C(|t_1 - t_2|)$ solves $(I - aD^2)C(|t_1 - t_2|) = b\delta(t)$, for suitable constants $a, b$.) Thus Proposition 2 generalizes a recent neuroscientific result: the likelihood for a certain neural model (the leaky integrate-and-fire model driven by Gaussian noise, for which the corresponding covariance is Ornstein-Uhlenbeck) is l.c. (*21, 22*) (of course, in this case the model was motivated by biophysical instead of learning-theoretic concerns).

In addition, multidimensional generalizations of this family are straightforward: corresponding kernels solve the Helmholtz problem,

$$(I - a\Delta)C(\vec{t}) = b\delta(\vec{t}),$$

with $\Delta$ denoting the Laplacian. Solutions to this problem are well-known: in the isotropic case, we obtain a family of radial Bessel functions, with $a, b$ again setting the overall magnitude and correlation scale of $C(\vec{t}_1, \vec{t}_2) = C(||\vec{t}_1 - \vec{t}_2||_2)$. Generalizing in a different direction, we could let $A_{\vec{\theta}}$ include higher-order differential terms, $A_{\vec{\theta}}^{-1} = \sum_{k=0} \theta_k D^k$; the resulting covariance kernels correspond to higher-order autoregression process priors.

An even broader class of kernel parameterizations may be developed in the spectral domain: still assuming stationary white noise inputs, we may diagonalize $C$ in the Fourier basis, that is, $C(\vec{\omega}) = O^t P(\vec{\omega}) O$, with $O$ the (unitary) Fourier transform operator and $P(\vec{\omega})$ the power spectral density. Thus, comparing to the conditions above, if the spectral density may be written as $P(\vec{\omega})^{-1} = |\sum_k \theta_k h_k(\vec{\omega})|^2$ (where $|.|$ denotes complex magnitude), for $\theta_k > 0$ and functions $h_k(\vec{\omega})$ such that $\text{sign}(\text{real}(h_k(\vec{\omega})))$ is constant in $k$ for any $\vec{\omega}$, then the likelihood will be l.c. in $\vec{\theta}$; $A_{\vec{\theta}}$ here may be taken as the multiplication operator

$O^t(\sum_k \theta_k h_k(\vec{\omega}))^{-1})$. Remember that the smoothness of the sample paths of $G(\vec{t})$ depends on the rate of decay of the spectral density (*1,23*); thus we may obtain smoother (or rougher) kernel families by choosing $\sum_k \theta_k h_k(\vec{\omega})$ as more rapidly- (or slowly-)increasing.

**Proofs**

*Predictive density.* This proof is a straightforward application of the Prekopa theorem (*10*). Write the predictive distributions as

$$p(\{L_k G\}|D) = K(D) \int p(\{L_k G\}, \{G(t_i), n(t_i)\}) p(\{y_i, t_i\}|\{L_k G\}, \{G(t_i), n(t_i)\}),$$

where $\{L_k\}$ is a finite set of continuous linear functionals of $G$, $K(D)$ is a constant that depends only on the data, the integral is over all $\{G(t_i), n(t_i)\}$, and $\{n_i, y_i\}$ is ignored in the unsupervised case. Now we need only prove that the multiplicands on the right hand side above are l.c. The log-concavity of the left term is assumed; the right term, in turn, can be rewritten as

$$p(\{y_i, t_i\}|\{L_k G\}, \{G(t_i), n(t_i)\}) = p(\{y_i, t_i\}|\{G(t_i), n(t_i)\}),$$

by the Markovian nature of the models. We prove the log-concavity of the right individually for each of our applications.

In the supervised cases, $\{t_i\}$ is given and so we only need to look at $p(\{y_i\}|\{G(t_i), n(t_i)\})$. In the classification case, this is simply an indicator of the set

$$\bigcap_i \left( G(t_i) + \sigma_i n_i \begin{cases} \leq 0, y_i = 0 \\ > 0, y_i = 1 \end{cases} \right),$$

which is jointly convex in $\{G(t_i), n(t_i)\}$, completing the proof in this case.

The regression case is proven in a similar fashion: write $p(\{y_i\}|\{G(t_i), n(t_i)\})$ as the limit as $\epsilon \to 0$ of the indicator of the convex set

$$\bigcap_i (|G(t_i) + \sigma_i n_i - y_i| < \epsilon),$$

then use the fact that pointwise limits preserve log-concavity. (The predictive distributions of $\{y(t)\}$ will also be l.c. here, by a nearly identical argument.)

In the density estimation case, the term

$$p(\{t_i\}|\{G(t_i)\}) = \prod_i G(t_i)$$

is obviously l.c. in $\{G(t_i)\}$. However, recall that we perturbed the distribution of $G(t)$ in this case as well, by conditioning $G(t)$ to be positive and normalized. The fact that $p(\{L_k G\}, \{G(t_i)\})$ is l.c. follows upon writing this term as a marginalization of densities which are products of l.c. densities with indicators of convex sets (enforcing the linear normalization and positivity constraints).

Finally, for the point process intensity case, write the likelihood term, as usual,

$$p(\{t_i\}|\{G(t_i)\}) = e^{-\int f(G(\vec{t}))d\vec{t}} \prod_i f(G(\vec{t_i})),$$

where $f$ is the scalar warping function that takes the original Gaussian function $G(\vec{t})$ into the space of intensity functions. This term is clearly l.c. whenever $f(s)$ is both convex and l.c. in $s$; for more details on this class of functions, see e.g. (*20*). $\square$

*Likelihood.* We begin with the unsupervised cases. In the density estimation case, write the likelihood as

$$L(\mu) = \int dp_\mu(G) 1_C(\{G(\vec{t})\}) \prod_i G(\vec{t_i}),$$

with $p_\mu(G)$ the probability of $G$ under $\mu$. Here $1_C$ is the (l.c.) indicator function of the convex set enforcing the linear constraints (positivity and normalization) on $G$. All three terms in the integrand on the right are clearly jointly l.c. in $(G, \mu)$. In the point process case,

$$L(\mu) = \int dp_\mu(G) e^{-\int f(G(\vec{t}))d\vec{t}} \prod_i f(G(\vec{t_i}));$$

the joint log-concavity of the three multiplicands on the right is again easily demonstrated.

The Prekopa theorem cannot be directly applied here, since the functions $1_C(.)$ and $e^{-\int f(.)}$ depend in an infinite-dimensional way on $G$ and $\mu$; however, we can apply the Prekopa theorem to any finite-dimensional approximation of these functions (e.g., by approximating the normalization condition and exponential integral by Riemann sums and the positivity condition at a finite number of points), then obtain the theorem in the limit as the approximation becomes infinitely fine, using the fact that pointwise limits preserve log-concavity.

For the supervised cases, write

$$
\begin{aligned}
L(\mu_L, \vec{\theta}, \{\sigma^{-1}\}) &= \lim_j \int dp_j(G_L, \vec{n}) 1\left(A_{j,\theta}(G_L + \mu_L^G) + \vec{\sigma}.(\vec{n} + \mu_L^n) \in V\right) \\
&= \lim_j \int dp_j(G_L, \vec{n}) 1\left((G_L, \vec{n}) \in (\sum_k \theta_k K_{j,k} V, \vec{\sigma}.^{-1}.V) + \mu_L\right),
\end{aligned}
$$

with $V$ an appropriate convex constraint set (or limit thereof) defined by the observed data $\{y_i\}$, $\mu_L^G$ and $\mu_L^n$ the projection of $\mu_L$ into $\mathcal{G}_j$ or $\Re^{Nd_G}$, respectively, and . denoting pointwise operations on vectors. The result now follows immediately from Rinott's theorem on convex translations of sets under l.c. probability measures (*11, 22*). $\qquad \square$

Again, we have not assumed anything more about $p(G_L, \vec{n})$ than log-concavity; as before, this allows dependence of $G$ and $\vec{n}$, anisotropic correlations, etc. It is worth noting, though, that the above result is somewhat stronger in the supervised case than the unsupervised; the proof of log-concavity in the covariance parameters $\vec{\theta}$ does not seem to generalize easily to the unsupervised setup (briefly, because $\log(\sum_k \theta_k y_k)$ is not jointly concave in $(\theta_k, y_k)$ for all $(\theta_k, y_k)$, $\theta_k y_k > 0$, precluding a direct application of the Prekopa or Rinott theorems in the unsupervised case). Extensions to ensure that the unsupervised likelihood is l.c. in $\vec{\theta}$ are possible, but require further restrictions on the form of $p(G|\vec{\theta})$ and will not be pursued here.

**Discussion**

We have provided some useful results on the log-concavity of the predictive densities and likelihoods associated with several common Gaussian process methods for machine learning. In particular, our results preclude the existence of non-global local maxima in these functions, for *any* observed data; moreover, Laplace approximations of these functions will not, in general, be disastrous, and efficient sampling methods are available.

Perhaps the main practical implication of our results stems from our proposition on the likelihood; we recommend a certain simple way to obtain parameterized families of kernels which respect this log-concavity property. Kernel families which may be obtained in this manner can range from extremely smooth to singular, and may model anisotropies

flexibly. Finally, these results indicate useful classes of constraints (or more generally, regularizing priors) on the hyperparameters; any prior which is l.c. (or any constraint set which is convex) in the parameterization discussed here will lead to l.c. *a posteriori* problems.

More generally, we have introduced some straightforward applications of a useful and interesting theorem. We expect that further applications in machine learning (e.g., in latent variable models, marginalization of hyperparameters, etc.) will be easy to find.

**Acknowledgements**: We thank Z. Ghahramani and C. Williams for many helpful conversations. LP is supported by an International Research Fellowship from the Royal Society.

## References

1. M. Seeger, *International Journal of Neural Systems* **14**, 1 (2004).

2. P. Sollich, A. Halees, *Neural Computation* **14**, 1393 (2002).

3. C. Williams, D. Barber, *IEEE PAMI* **20**, 1342 (1998).

4. M. Gibbs, D. MacKay, *IEEE Transactions on Neural Networks* **11**, 1458 (2000).

5. L. Csato, Gaussian processes - iterative sparse approximations, Ph.D. thesis, Aston U. (2002).

6. T. Minka, A family of algorithms for approximate bayesian inference, Ph.D. thesis, MIT (2001).

7. W. Gilks, P. Wild, *Applied Statistics* **41**, 337 (1992).

8. R. Neal, *Annals of Statistics* **31**, 705 (2003).

9. L. Lovasz, S. Vempala, The geometry of logconcave functions and an $O^*(n^3)$ sampling algorithm, *Tech. Rep. 2003-04*, Microsoft Research (2003).

10. A. Prekopa, *Acad Sci. Math.* **34**, 335 (1973).

11. Y. Rinott, *Annals of Probability* **4**, 1020 (1976).

12. P. McCullagh, J. Nelder, *Generalized linear models* (Chapman and Hall, London, 1989).

13. J. Oakley, A. O'Hagan, *Biometrika* **under review** (2003).

14. I. Good, R. Gaskins, *Biometrika* **58**, 255 (1971).

15. W. Bialek, C. Callan, S. Strong, *Physical Review Letters* **77**, 4693 (1996).

16. D. Snyder, M. Miller, *Random Point Processes in Time and Space* (Springer-Verlag, 1991).

17. J. Moller, A. Syversveen, R. Waagepetersen, *Scandinavian Journal of Statistics* **25**, 451 (1998).

18. I. DiMatteo, C. Genovese, R. Kass, *Biometrika* **88**, 1055 (2001).

19. R. Neal, Monte Carlo implementation of Gaussian process models for Bayesian regression and classification, *Tech. Rep. 9702*, University of Toronto (1997).

20. L. Paninski, *Network: Computation in Neural Systems* **15**, 243 (2004).

21. J. Pillow, L. Paninski, E. Simoncelli, *NIPS* **17** (2003).

22. L. Paninski, J. Pillow, E. Simoncelli, *Neural Computation* **16**, 2533 (2004).

23. H. Dym, H. McKean, *Fourier Series and Integrals* (Academic Press, New York, 1972).
